# Monte-Carlo Planning in Large POMDPs

**David Silver**
MIT, Cambridge, MA 02139
davidstarsilver@gmail.com

**Joel Veness**
UNSW, Sydney, Australia
jveness@gmail.com

## Abstract

This paper introduces a Monte-Carlo algorithm for online planning in large POMDPs. The algorithm combines a Monte-Carlo update of the agent's belief state with a Monte-Carlo tree search from the current belief state. The new algorithm, *POMCP*, has two important properties. First, Monte-Carlo sampling is used to break the curse of dimensionality both during belief state updates and during planning. Second, only a black box simulator of the POMDP is required, rather than explicit probability distributions. These properties enable POMCP to plan effectively in significantly larger POMDPs than has previously been possible. We demonstrate its effectiveness in three large POMDPs. We scale up a well-known benchmark problem, *rocksample*, by several orders of magnitude. We also introduce two challenging new POMDPs: $10 \times 10$ *battleship* and *partially observable PacMan*, with approximately $10^{18}$ and $10^{56}$ states respectively. Our Monte-Carlo planning algorithm achieved a high level of performance with no prior knowledge, and was also able to exploit simple domain knowledge to achieve better results with less search. POMCP is the first general purpose planner to achieve high performance in such large and unfactored POMDPs.

## 1 Introduction

*Monte-Carlo tree search* (MCTS) is a new approach to online planning that has provided exceptional performance in large, fully observable domains. It has outperformed previous planning approaches in challenging games such as Go [5], Amazons [10] and General Game Playing [4]. The key idea is to evaluate each state in a search tree by the average outcome of simulations from that state. MCTS provides several major advantages over traditional search methods. It is a highly selective, best-first search that quickly focuses on the most promising regions of the search space. It breaks the curse of dimensionality by sampling state transitions instead of considering all possible state transitions. It only requires a black box simulator, and can be applied in problems that are too large or too complex to represent with explicit probability distributions. It uses random simulations to estimate the potential for long-term reward, so that it plans over large horizons, and is often effective without any search heuristics or prior domain knowledge [8]. If exploration is controlled appropriately then MCTS converges to the optimal policy. In addition, it is anytime, computationally efficient, and highly parallelisable.

In this paper we extend MCTS to *partially observable* environments (POMDPs). Full-width planning algorithms, such as value iteration [6], scale poorly for two reasons, sometimes referred to as the *curse of dimensionality* and the *curse of history* [12]. In a problem with $n$ states, value iteration reasons about an $n$-dimensional belief state. Furthermore, the number of histories that it must evaluate is exponential in the horizon. The basic idea of our approach is to use Monte-Carlo sampling to break *both* curses, by sampling start states from the belief state, and by sampling histories using a black box simulator.

Our search algorithm constructs, online, a search tree of histories. Each node of the search tree estimates the value of a history by Monte-Carlo simulation. For each simulation, the

start state is sampled from the current belief state, and state transitions and observations are sampled from a black box simulator. We show that if the belief state is correct, then this simple procedure converges to the optimal policy for any finite horizon POMDP. In practice we can execute hundreds of thousands of simulations per second, which allows us to construct extensive search trees that cover many possible contingencies. In addition, Monte-Carlo simulation can be used to update the agent's belief state. As the search tree is constructed, we store the set of sample states encountered by the black box simulator in each node of the search tree. We approximate the belief state by the set of sample states corresponding to the actual history. Our algorithm, *Partially Observable Monte-Carlo Planning* (POMCP), efficiently uses the same set of Monte-Carlo simulations for both tree search and belief state updates.

## 2 Background

### 2.1 POMDPs

In a Markov decision process (MDP) the environment's dynamics are fully determined by its current state $s_t$. For any state $s \in \mathcal{S}$ and for any action $a \in \mathcal{A}$, the *transition probabilities* $\mathcal{P}_{ss'}^a = Pr(s_{t+1} = s'|s_t = s, a_t = a)$ determine the next state distribution $s'$, and the *reward function* $\mathcal{R}_s^a = \mathbb{E}[r_{t+1}|s_t = s, a_t = a]$ determines the expected reward. In a partially observable Markov decision process (POMDP), the state cannot be directly observed by the agent. Instead, the agent receives an observation $o \in \mathcal{O}$, determined by *observation probabilities* $\mathcal{Z}_{s'o}^a = Pr(o_{t+1} = o|s_{t+1} = s', a_t = a)$. The initial state $s_0 \in \mathcal{S}$ is determined by a probability distribution $\mathcal{I}_s = Pr(s_0 = s)$. A *history* is a sequence of actions and observations, $h_t = \{a_1, o_1, ..., a_t, o_t\}$ or $h_t a_{t+1} = \{a_1, o_1, ..., a_t, o_t, a_{t+1}\}$. The agent's action-selection behaviour can be described by a *policy*, $\pi(h, a)$, that maps a history $h$ to a probability distribution over actions, $\pi(h, a) = Pr(a_{t+1} = a|h_t = h)$. The *return* $R_t = \sum_{k=t}^{\infty} \gamma^{k-t} r_k$ is the total discounted reward accumulated from time $t$ onwards, where $\gamma$ is a discount factor specified by the environment. The *value function* $V^\pi(h)$ is the expected return from state $s$ when following policy $\pi$, $V^\pi(h) = \mathbb{E}_\pi[R_t|h_t = h]$. The *optimal value function* is the maximum value function achievable by any policy, $V^*(h) = \max_\pi V^\pi(h)$. In any POMDP there is at least one *optimal policy* $\pi^*(h, a)$ that achieves the optimal value function. The *belief state* is the probability distribution over states given history $h$, $\mathcal{B}(s, h) = Pr(s_t = s|h_t = h)$.

### 2.2 Online Planning in POMDPs

Online POMDP planners use forward search, from the current history or belief state, to form a local approximation to the optimal value function. The majority of online planners are based on point-based value iteration [12, 13]. These algorithms use an explicit model of the POMDP probability distributions, $\mathcal{M} = \langle \mathcal{P}, \mathcal{R}, \mathcal{Z}, \mathcal{I} \rangle$. They construct a search tree of belief states, using a heuristic best-first expansion procedure. Each value in the search tree is updated by a full-width computation that takes account of all possible actions, observations and next states. This approach can be combined with an offline planning method to produce a branch-and-bound procedure [13]. Upper or lower bounds on the value function are computed offline, and are propagated up the tree during search. If the POMDP is small, or can be factored into a compact representation, then full-width planning with explicit models can be very effective.

Monte-Carlo planning is a very different paradigm for online planning in POMDPs [2, 7]. The agent uses a simulator $\mathcal{G}$ as a *generative* model of the POMDP. The simulator provides a sample of a successor state, observation and reward, given a state and action, $(s_{t+1}, o_{t+1}, r_{t+1}) \sim \mathcal{G}(s_t, a_t)$, and can also be reset to a start state $s$. The simulator is used to generate sequences of states, observations and rewards. These simulations are used to update the value function, without ever looking inside the black box describing the model's dynamics. In addition, Monte-Carlo methods have a sample complexity that is determined only by the underlying difficulty of the POMDP, rather than the size of the state space or observation space [7]. In principle, this makes them an appealing choice for large POMDPs. However, prior Monte-Carlo planners have been limited to fixed horizon, depth-first search [7] (also known as *sparse sampling*), or to simple rollout methods with no search tree [2], and have not so far proven to be competitive with best-first, full-width planning methods.

## 2.3  Rollouts

In fully observable MDPs, Monte-Carlo simulation provides a simple method for evaluating a state $s$. Sequences of states are generated by an MDP simulator, starting from $s$ and using a random rollout policy, until a terminal state or discount horizon is reached. The value of state $s$ is estimated by the mean return of $N$ simulations from $s$, $V(s) = \frac{1}{N}\sum_{i=1}^{N} R^i$, where $R^i$ is the return from the beginning of the $i$th simulation. Monte-Carlo simulation can be turned into a simple control algorithm by evaluating all legal actions and selecting the action with highest evaluation [15]. Monte-Carlo simulation can be extended to partially observable MDPs [2] by using a history based rollout policy $\pi_{rollout}(h, a)$. To evaluate candidate action $a$ in history $h$, simulations are generated from $ha$ using a POMDP simulator and the rollout policy. The value of $ha$ is estimated by the mean return of $N$ simulations from $ha$.

## 2.4  Monte-Carlo Tree Search

*Monte-Carlo tree search* [3] uses Monte-Carlo simulation to evaluate the nodes of a search tree in a sequentially best-first order. There is one node in the tree for each state $s$, containing a value $Q(s, a)$ and a visitation count $N(s, a)$ for each action $a$, and an overall count $N(s) = \sum_a N(s, a)$. Each node is initialised to $Q(s, a) = 0, N(s, a) = 0$. The value is estimated by the mean return from $s$ of all simulations in which action $a$ was selected from state $s$. Each simulation starts from the current state $s_t$, and is divided into two stages: a *tree policy* that is used while within the search tree; and a rollout policy that is used once simulations leave the scope of the search tree. The simplest version of MCTS uses a greedy tree policy during the first stage, which selects the action with the highest value; and a uniform random rollout policy during the second stage. After each simulation, one new node is added to the search tree, containing the first state visited in the second stage. The UCT algorithm [8] improves the greedy action selection in MCTS. Each state of the search tree is viewed as a multi-armed bandit, and actions are chosen by using the UCB1 algorithm [1]. The value of an action is augmented by an exploration bonus that is highest for rarely tried actions, $Q^\oplus(s, a) = Q(s, a) + c\sqrt{\frac{\log N(s)}{N(s,a)}}$. The scalar constant $c$ determines the relative ratio of exploration to exploitation; when $c = 0$ the UCT algorithm acts greedily within the tree. Once all actions from state $s$ are represented in the search tree, the tree policy selects the action maximising the augmented action-value, $\operatorname{argmax}_a Q^\oplus(s, a)$. Otherwise, the rollout policy is used to select actions. For suitable choice of $c$, the value function constructed by UCT converges in probability to the optimal value function, $Q(s, a) \xrightarrow{p} Q^*(s, a), \forall s \in \mathcal{S}, a \in \mathcal{A}$ [8]. UCT can be extended to use domain knowledge, for example heuristic knowledge or a value function computed offline [5]. New nodes are initialised using this knowledge, $Q(s, a) = Q_{init}(s, a), N(s, a) = N_{init}$, where $Q_{init}(s, a)$ is an action value function and $N_{init}$ indicates its quality. Domain knowledge narrowly focuses the search on promising states without altering asymptotic convergence.

# 3  Monte-Carlo Planning in POMDPs

Partially Observable Monte-Carlo Planning (POMCP) consists of a UCT search that selects actions at each time-step; and a particle filter that updates the agent's belief state.

## 3.1  Partially Observable UCT (PO–UCT)

We extend the UCT algorithm to partially observable environments by using a search tree of histories instead of states. The tree contains a node $T(h) = \langle N(h), V(h) \rangle$ for each represented history $h$. $N(h)$ counts the number of times that history $h$ has been visited. $V(h)$ is the value of history $h$, estimated by the mean return of all simulations starting with $h$. New nodes are initialised to $\langle V_{init}(h), N_{init}(h) \rangle$ if domain knowledge is available, and to $\langle 0, 0 \rangle$ otherwise. We assume for now that the belief state $\mathcal{B}(s, h)$ is known exactly. Each simulation starts in an initial state that is sampled from $\mathcal{B}(\cdot, h_t)$. As in the fully observable algorithm, the simulations are divided into two stages. In the first stage of simulation, when child nodes exist for all children, actions are selected by UCB1, $V^\oplus(ha) = V(ha) + c\sqrt{\frac{\log N(h)}{N(ha)}}$. Actions are then selected to maximise this augmented value, $\operatorname{argmax}_a V^\oplus(ha)$. In the second

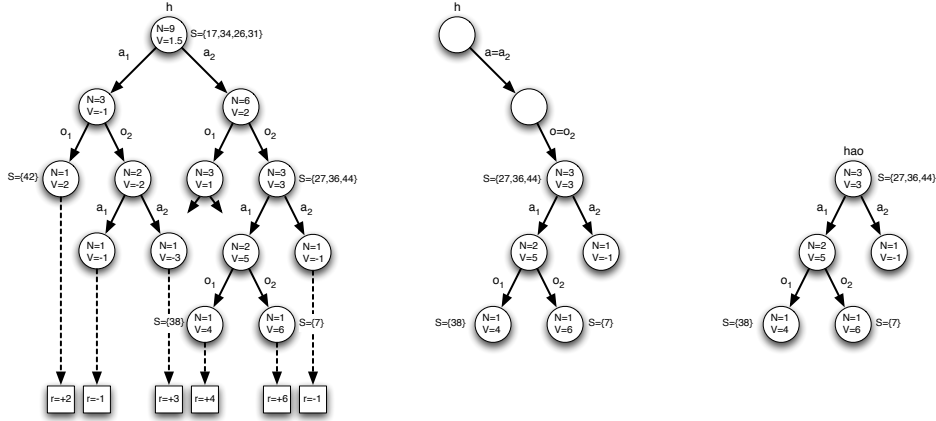

Figure 1: An illustration of POMCP in an environment with 2 actions, 2 observations, 50 states, and no intermediate rewards. The agent constructs a search tree from multiple simulations, and evaluates each history by its mean return (left). The agent uses the search tree to select a real action $a$, and observes a real observation $o$ (middle). The agent then prunes the tree and begins a new search from the updated history $hao$ (right).

stage of simulation, actions are selected by a history based rollout policy $\pi_{rollout}(h, a)$ (e.g. uniform random action selection). After each simulation, precisely one new node is added to the tree, corresponding to the first new history encountered during that simulation.

## 3.2 Monte-Carlo Belief State Updates

In small state spaces, the belief state can be updated exactly by Bayes' theorem, $\mathcal{B}(s', hao) = \frac{\sum_{s \in \mathcal{S}} \mathcal{Z}^a_{s'o} \mathcal{P}^a_{ss'} \mathcal{B}(s,h)}{\sum_{s \in \mathcal{S}} \sum_{s'' \in \mathcal{S}} \mathcal{Z}^a_{s''o} \mathcal{P}^a_{ss''} \mathcal{B}(s,h)}$. The majority of POMDP planning methods operate in this manner [13]. However, in large state spaces even a single Bayes update may be computationally infeasible. Furthermore, a compact representation of the transition or observation probabilities may not be available. To plan efficiently in large POMDPs, we approximate the belief state using an unweighted particle filter, and use a Monte-Carlo procedure to update particles based on sample observations, rewards, and state transitions. Although weighted particle filters are used widely to represent belief states, an unweighted particle filter can be implemented particularly efficiently with a black box simulator, without requiring an explicit model of the POMDP, and providing excellent scalability to larger problems.

We approximate the belief state for history $h_t$ by $K$ particles, $B_t^i \in \mathcal{S}, 1 \leq i \leq K$. Each particle corresponds to a sample state, and the belief state is the sum of all particles, $\hat{\mathcal{B}}(s, h_t) = \frac{1}{K} \sum_{i=1}^{K} \delta_{sB_t^i}$, where $\delta_{ss'}$ is the kronecker delta function. At the start of the algorithm, $K$ particles are sampled from the initial state distribution, $B_0^i \sim \mathcal{I}, 1 \leq i \leq K$. After a real action $a_t$ is executed, and a real observation $o_t$ is observed, the particles are updated by Monte-Carlo simulation. A state $s$ is sampled from the current belief state $\hat{\mathcal{B}}(s, h_t)$, by selecting a particle at random from $B_t$. This particle is passed into the black box simulator, to give a successor state $s'$ and observation $o'$, $(s', o', r) \sim \mathcal{G}(s, a_t)$. If the sample observation matches the real observation, $o = o_t$, then a new particle $s'$ is added to $B_{t+1}$. This process repeats until $K$ particles have been added. This approximation to the belief state approaches the true belief state with sufficient particles, $\lim_{K \to \infty} \hat{\mathcal{B}}(s, h_t) = \mathcal{B}(s, h_t), \forall s \in \mathcal{S}$. As with many particle filter approaches, particle deprivation is possible for large $t$. In practice we combine the belief state update with particle reinvigoration. For example, new particles can be introduced by adding artificial noise to existing particles.

## 3.3 Partially Observable Monte-Carlo

POMCP combines Monte-Carlo belief state updates with PO–UCT, and shares the same simulations for both Monte-Carlo procedures. Each node in the search tree, $T(h) = \langle N(h), V(h), B(h) \rangle$, contains a set of particles $B(h)$ in addition to its count $N(h)$ and value $V(h)$. The search procedure is called from the current history $h_t$. Each simulation begins from a start state that is sampled from the belief state $B(h_t)$. Simulations are performed

---

**Algorithm 1** Partially Observable Monte-Carlo Planning

| | |
|---|---|
| **procedure** SEARCH($h$) | **procedure** SIMULATE($s, h, depth$) |
|   **repeat** |   **if** $\gamma^{depth} < \epsilon$ **then** |
|     **if** $h = empty$ **then** |     **return** 0 |
|       $s \sim \mathcal{I}$ |   **end if** |
|     **else** |   **if** $h \notin T$ **then** |
|       $s \sim B(h)$ |     **for all** $a \in \mathcal{A}$ **do** |
|     **end if** |       $T(ha) \leftarrow (N_{init}(ha), V_{init}(ha), \emptyset)$ |
|     SIMULATE($s, h, 0$) |     **end for** |
|   **until** TIMEOUT() |     **return** ROLLOUT($s, h, depth$) |
|   **return** $\underset{b}{\mathrm{argmax}}\ V(hb)$ |   **end if** |
| **end procedure** | $a \leftarrow \underset{b}{\mathrm{argmax}}\ V(hb) + c\sqrt{\frac{\log N(h)}{N(hb)}}$ |
| | $(s', o, r) \sim \mathcal{G}(s, a)$ |
| **procedure** ROLLOUT($s, h, depth$) | $R \leftarrow r + \gamma.\text{SIMULATE}(s', hao, depth + 1)$ |
|   **if** $\gamma^{depth} < \epsilon$ **then** | $B(h) \leftarrow B(h) \cup \{s\}$ |
|     **return** 0 | $N(h) \leftarrow N(h) + 1$ |
|   **end if** | $N(ha) \leftarrow N(ha) + 1$ |
|   $a \sim \pi_{rollout}(h, \cdot)$ | $V(ha) \leftarrow V(ha) + \frac{R - V(ha)}{N(ha)}$ |
|   $(s', o, r) \sim \mathcal{G}(s, a)$ |   **return** $R$ |
|   **return** $r + \gamma.\text{ROLLOUT}(s', hao, depth+1)$ | **end procedure** |
| **end procedure** | |

---

using the partially observable UCT algorithm, as described above. For every history $h$ encountered during simulation, the belief state $B(h)$ is updated to include the simulation state. When search is complete, the agent selects the action $a_t$ with greatest value, and receives a real observation $o_t$ from the world. At this point, the node $T(h_t a_t o_t)$ becomes the root of the new search tree, and the belief state $B(h_t ao)$ determines the agent's new belief state. The remainder of the tree is pruned, as all other histories are now impossible. The complete POMCP algorithm is described in Algorithm 1 and Figure 1.

## 4 Convergence

The UCT algorithm converges to the optimal value function in fully observable MDPs [8]. This suggests two simple ways to apply UCT to POMDPs: either by converting every belief state into an MDP state, or by converting every history into an MDP state, and then applying UCT directly to the derived MDP. However, the first approach is computationally expensive in large POMDPs, where even a single belief state update can be prohibitively costly. The second approach requires a history-based simulator that can sample the next history given the current history, which is usually more costly and hard to encode than a state-based simulator. The key innovation of the PO–UCT algorithm is to apply a UCT search to a history-based MDP, but using a state-based simulator to efficiently sample states from the current beliefs. In this section we prove that given the true belief state $\mathcal{B}(s, h)$, PO–UCT also converges to the optimal value function. We prove convergence for POMDPs with finite horizon $T$; this can be extended to the infinite horizon case as suggested in [8].

**Lemma 1.** *Given a POMDP $\mathcal{M} = (\mathcal{S}, \mathcal{A}, \mathcal{P}, \mathcal{R}, \mathcal{Z})$, consider the derived MDP with histories as states, $\tilde{\mathcal{M}} = (\mathcal{H}, \mathcal{A}, \tilde{\mathcal{P}}, \tilde{\mathcal{R}})$, where $\tilde{\mathcal{P}}^a_{h,hao} = \sum_{s \in \mathcal{S}} \sum_{s' \in \mathcal{S}} \mathcal{B}(s, h) \mathcal{P}^a_{ss'} \mathcal{Z}^a_{s'o}$ and $\tilde{\mathcal{R}}^a_h = \sum_{s \in \mathcal{S}} \mathcal{B}(s, h) \mathcal{R}^a_s$. Then the value function $\tilde{V}^\pi(h)$ of the derived MDP is equal to the value function $V^\pi(h)$ of the POMDP, $\forall \pi\ \tilde{V}^\pi(h) = V^\pi(h)$.*

*Proof.* By backward induction on the Bellman equation, starting from the horizon, $V^\pi(h) = \sum_{s \in \mathcal{S}} \sum_{a \in \mathcal{A}} \sum_{s' \in \mathcal{S}} \sum_{o \in \mathcal{O}} \mathcal{B}(s, h) \pi(h, a) \left( \mathcal{R}^a_s + \gamma \mathcal{P}^a_{ss'} \mathcal{Z}^a_{s'o} V^\pi(hao) \right) = \sum_{a \in \mathcal{A}} \sum_{o \in \mathcal{O}} \pi(h, a) \left( \tilde{\mathcal{R}}^a_h + \gamma \tilde{P}^a_{h,hao} \tilde{V}^\pi(hao) \right) = \tilde{V}^\pi(h)$. $\qquad \square$

Let $\mathcal{D}^\pi(h_T)$ be the *POMDP rollout distribution*. This is the distribution of histories generated by sampling an initial state $s_t \sim \mathcal{B}(s, h_t)$, and then repeatedly sampling actions from policy $\pi(h, a)$ and sampling states, observations and rewards from $\mathcal{M}$, until terminating at

time $T$. Let $\tilde{\mathcal{D}}^\pi(h_T)$ be the *derived MDP rollout distribution*. This is the distribution of histories generated by starting at $h_t$ and then repeatedly sampling actions from policy $\pi$ and sampling state transitions and rewards from $\tilde{\mathcal{M}}$, until terminating at time $T$.

**Lemma 2.** *For any rollout policy $\pi$, the POMDP rollout distribution is equal to the derived MDP rollout distribution, $\forall \pi\ \mathcal{D}^\pi(h_T) = \tilde{\mathcal{D}}^\pi(h_T)$.*

*Proof.* By forward induction from $h_t$, $\mathcal{D}^\pi(hao) = \mathcal{D}^\pi(h)\pi(h,a)\sum_{s\in\mathcal{S}}\sum_{s'\in\mathcal{S}}\mathcal{B}(s,h)\mathcal{P}^a_{ss'}\mathcal{Z}^a_{s'o} = \tilde{\mathcal{D}}^\pi(h)\pi(h,a)\tilde{\mathcal{P}}^a_{h,hao} = \tilde{\mathcal{D}}^\pi(hao)$. $\qquad\square$

**Theorem 1.** *For suitable choice of $c$, the value function constructed by PO–UCT converges in probability to the optimal value function, $V(h) \xrightarrow{p} V^*(h)$, for all histories $h$ that are prefixed by $h_t$. As the number of visits $N(h)$ approaches infinity, the bias of the value function, $\mathbb{E}\left[V(h) - V^*(h)\right]$ is $O(\log N(h)/N(h))$.*

*Proof.* By Lemma 2 the PO–UCT simulations can be mapped into UCT simulations in the derived MDP. By Lemma 1, the analysis of UCT in [8] can then be applied to PO–UCT. $\quad\square$

## 5 Experiments

We applied POMCP to the benchmark *rocksample* problem, and to two new problems: *battleship* and *pocman*. For each problem we ran POMCP 1000 times, or for up to 12 hours of total computation time. We evaluated the performance of POMCP by the average total discounted reward. In the smaller *rocksample* problems, we compared POMCP to the best full-width online planning algorithms. However, the other problems were too large to run these algorithms. To provide a performance benchmark in these cases, we evaluated the performance of simple Monte-Carlo simulation without any tree. The *PO-rollout algorithm* used Monte-Carlo belief state updates, as described in section 3.2. It then simulated $n/|\mathcal{A}|$ rollouts for each legal action, and selected the action with highest average return.

The exploration constant for POMCP was set to $c = R_{hi} - R_{lo}$, where $R_{hi}$ was the highest return achieved during sample runs of POMCP with $c = 0$, and $R_{lo}$ was the lowest return achieved during sample rollouts. The discount horizon was set to 0.01 (about 90 steps when $\gamma = 0.95$). On the larger *battleship* and *pocman* problems, we combined POMCP with particle reinvigoration. After each real action and observation, additional particles were added to the belief state, by applying a domain specific local transformation to existing particles. When $n$ simulations were used, $n/16$ new particles were added to the belief set. We also introduced domain knowledge into the search algorithm, by defining a set of *preferred actions* $\mathcal{A}_p$. In each problem, we applied POMCP both with and without preferred actions. When preferred actions were used, the rollout policy selected actions uniformly from $\mathcal{A}_p$, and each new node $T(ha)$ in the tree was initialised to $V_{init}(ha) = R_{hi}, N_{init}(ha) = 10$ for preferred actions $a \in \mathcal{A}_p$, and to $V_{init}(ha) = R_{lo}, N_{init}(ha) = 0$ for all other actions. Otherwise, the rollouts policy selected actions uniformly among all legal actions, and each new node $T(ha)$ was initialised to $V_{init}(ha) = 0, N_{init}(ha) = 0$ for all $a \in \mathcal{A}$.

The *rocksample* $(n,k)$ problem [14] simulates a Mars explorer robot in an $n \times n$ grid containing $k$ rocks. The task is to determine which rocks are valuable, take samples of valuable rocks, and leave the map to the east when sampling is complete. When provided with an exactly factored representation, online full-width planners have been successful in *rocksample* $(7,8)$ [13], and an offline full-width planner has been successful in the much larger *rocksample* $(11,11)$ problem [11]. We applied POMCP to three variants of *rocksample*: $(7,8)$, $(11,11)$, and $(15,15)$, without factoring the problem. When using preferred actions, the number of valuable and unvaluable observations was counted for each rock. Actions that sampled rocks with more valuable observations were preferred. If all remaining rocks had a greater number of unvaluable observations, then the *east* action was preferred. The results of applying POMCP to *rocksample*, with various levels of prior knowledge, is shown in Figure 2. These results are compared with prior work in Table 1. On *rocksample* $(7,8)$, the performance of POMCP with preferred actions was close to the best prior online planning methods combined with offline solvers. On *rocksample* $(11,11)$, POMCP achieved the same performance with 4 seconds of online computation to the state-of-the-art solver SARSOP with 1000 seconds of offline computation [11]. Unlike prior methods, POMCP also provided scalable performance on *rocksample* $(15,15)$, a problem with over 7 million states.

| Rocksample | (7, 8) | (11, 11) | (15, 15) |
|---|---|---|---|
| States $\|\mathcal{S}\|$ | 12,544 | 247,808 | 7,372,800 |
| AEMS2 | 21.37 ±0.22 | N/A | N/A |
| HSVI-BFS | 21.46 ±0.22 | N/A | N/A |
| SARSOP | 21.39 ±0.01 | 21.56 ±0.11 | N/A |
| Rollout | 9.46 ±0.27 | 8.70 ±0.29 | 7.56 ±0.25 |
| POMCP | 20.71 ±0.21 | 20.01 ±0.23 | 15.32 ±0.28 |

Table 1: Comparison of Monte-Carlo planning with full-width planning on *rocksample*. POMCP and the rollout algorithm used prior knowledge in their rollouts. The online planners used knowledge computed offline by PBVI; results are from [13]. Each online algorithm was given 1 second per action. The full-width, offline planner SARSOP was given approximately 1000 seconds of offline computation; results are from [9]. All full-width planners were provided with an exactly factored representation of the POMDP; the Monte-Carlo planners do not factor the representation. The full-width planners could not be run on the larger problems.

In the *battleship* POMDP, 5 ships are placed at random into a $10 \times 10$ grid, subject to the constraint that no ship may be placed adjacent or diagonally adjacent to another ship. Each ship has a different size of $5 \times 1$, $4 \times 1$, $3 \times 1$ and $2 \times 1$ respectively. The goal is to find and sink all ships. However, the agent cannot observe the location of the ships. Each step, the agent can fire upon one cell of the grid, and receives an observation of 1 if a ship was hit, and 0 otherwise. There is a -1 reward per time-step, a terminal reward of +100 for hitting every cell of every ship, and there is no discounting ($\gamma = 1$). It is illegal to fire twice on the same cell. If it was necessary to fire on all cells of the grid, the total reward is 0; otherwise the total reward indicates the number of steps better than the worst case. There are 100 actions, 2 observations, and approximately $10^{18}$ states in this challenging POMDP. Particle invigoration is particularly important in this problem. Each local transformation applied one of three transformations: 2 ships of different sizes swapped location; 2 smaller ships were swapped into the location of 1 larger ship; or 1 to 4 ships were moved to a new location, selected uniformly at random, and accepted if the new configuration was legal. Without preferred actions, all legal actions were considered. When preferred actions were used, impossible cells for ships were deduced automatically, by marking off the diagonally adjacent cells to each hit. These cells were never selected in the tree or during rollouts. The performance of POMCP, with and without preferred actions, is shown in Figure 2. POMCP was able to sink all ships more than 50 moves faster, on average, than random play, and more than 25 moves faster than randomly selecting amongst preferred actions (which corresponds to the simple strategy used by many humans when playing the Battleship game). Using preferred actions, POMCP achieved better results with less search; however, even without preferred actions, POMCP was able to deduce the diagonal constraints from its rollouts, and performed almost as well given more simulations per move. Interestingly, the search tree only provided a small benefit over the PO-rollout algorithm, due to small differences between the value of actions, but high variance in the returns.

In our final experiment we introduce a partially observable version of the video game Pac-Man. In this task, *pocman*, the agent must navigate a $17 \times 19$ maze and eat the food pellets that are randomly distributed across the maze. Four ghosts roam the maze, initially according to a randomised strategy: at each junction point they select a direction, without doubling back, with probability proportional to the number of food pellets in line of sight in that direction. Normally, if PocMan touches a ghost then he dies and the episode terminates. However, four power pills are available, which last for 15 steps and enable PocMan to eat any ghosts he touches. If a ghost is within Manhattan distance of 5 of PocMan, it chases him aggressively, or runs away if he is under the effect of a power pill. The PocMan agent receives a reward of $-1$ at each step, $+10$ for each food pellet, $+25$ for eating a ghost and $-100$ for dying. The discount factor is $\gamma = 0.95$. The PocMan agent receives ten observation bits at every time step, corresponding to his senses of sight, hearing, touch and smell. He receives four observation bits indicating whether he can *see* a ghost in each cardinal direction, set to 1 if there is a ghost in his direct line of sight. He receives one observation bit indicating whether he can *hear* a ghost, which is set to 1 if he is within Manhattan distance 2 of a ghost. He receives four observation bits indicating whether he can *feel* a wall in each of the cardinal directions, which is set to 1 if he is adjacent to a wall. Finally, he receives one observation bit indicating whether he can *smell* food, which is set to 1 if he is adjacent or diagonally ad-

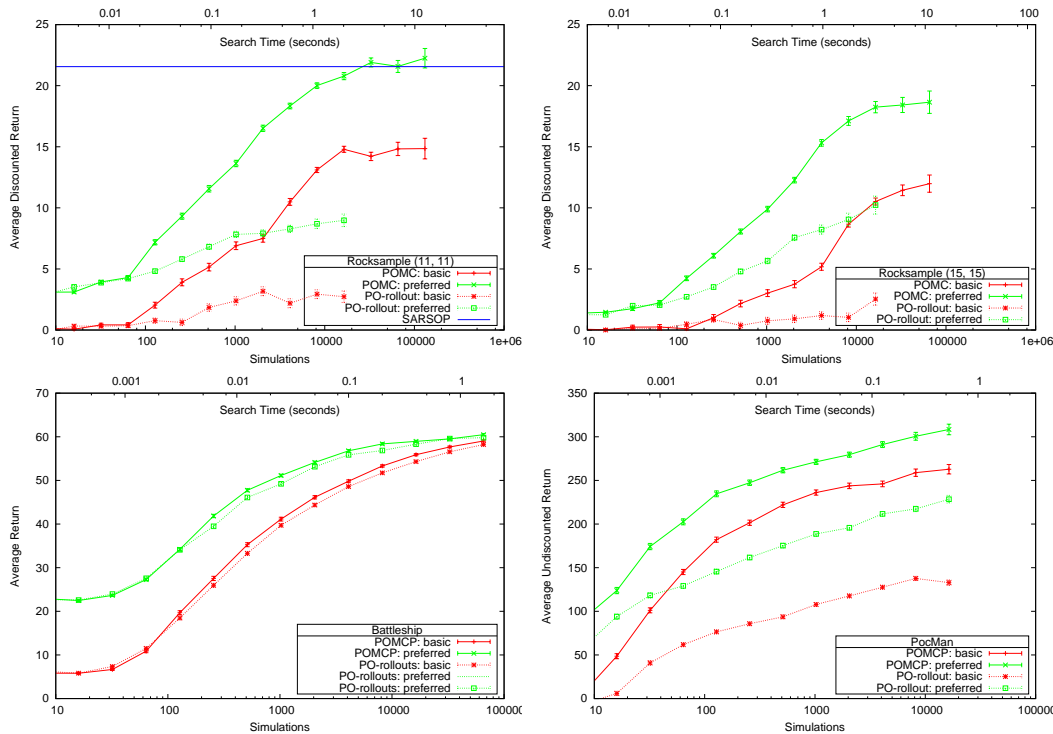

Figure 2: Performance of POMCP in *rocksample* (11,11) and (15,15), *battleship* and *pocman*. Each point shows the mean discounted return from 1000 runs or 12 hours of total computation. The search time for POMCP with preferred actions is shown on the top axis.

jacent to any food. The *pocman* problem has approximately $10^{56}$ states, 4 actions, and 1024 observations. For particle invigoration, 1 or 2 ghosts were teleported to a randomly selected new location. The new particle was accepted if consistent with the last observation. When using preferred actions, if PocMan was under the effect of a power pill, then he preferred to move in directions where he saw ghosts. Otherwise, PocMan preferred to move in directions where he didn't see ghosts, excluding the direction he just came from. The performance of POMCP in *pocman*, with and without preferred actions, is shown in Figure 2. Using preferred actions, POMCP achieved an average undiscounted return of over 300, compared to 230 for the PO-rollout algorithm. Without domain knowledge, POMCP still achieved an average undiscounted return of 260, compared to 130 for simple rollouts. A real-time demonstration of POMCP using preferred actions, is available online, along with source code for POMCP (http://www.cs.ucl.ac.uk/staff/D.Silver/web/Applications.html).

# 6 Discussion

Traditionally, POMDP planning has focused on small problems that have few states or can be neatly factorised into a compact representation. However, real-world problems are often large and messy, with enormous state spaces and probability distributions that cannot be conveniently factorised. In these challenging POMDPs, Monte-Carlo simulation provides an effective mechanism both for tree search and for belief state updates, breaking the curse of dimensionality and allowing much greater scalability than has previously been possible. Unlike previous approaches to Monte-Carlo planning in POMDPs, the PO–UCT algorithm provides a computationally efficient best-first search that focuses its samples in the most promising regions of the search space. The POMCP algorithm uses these same samples to provide a rich and effective belief state update. The *battleship* and *pocman* problems provide two examples of large POMDPs which cannot easily be factored and are intractable to prior algorithms for POMDP planning. POMCP was able to achieve high performance in these challenging problems with just a few seconds of online computation.

# References

[1] P. Auer, N. Cesa-Bianchi, and P. Fischer. Finite-time analysis of the multi-armed bandit problem. *Machine Learning*, 47(2-3):235–256, 2002.

[2] D. Bertsekas and D. Castañon. Rollout algorithms for stochastic scheduling problems. *Journal of Heuristics*, 5(1):89–108, 1999.

[3] R. Coulom. Efficient selectivity and backup operators in Monte-Carlo tree search. In *5th International Conference on Computer and Games, 2006-05-29*, pages 72–83, 2006.

[4] H. Finnsson and Y. Björnsson. Simulation-based approach to general game playing. In *23rd Conference on Artificial Intelligence*, pages 259–264, 2008.

[5] S. Gelly and D. Silver. Combining online and offline learning in UCT. In *17th International Conference on Machine Learning*, pages 273–280, 2007.

[6] L. Kaelbling, M. Littman, and A. Cassandra. Planning and acting in partially observable stochastic domains. *Artificial Intelligence*, 101:99–134, 1995.

[7] M. Kearns, Y. Mansour, and A. Ng. Approximate planning in large POMDPs via reusable trajectories. In *Advances in Neural Information Processing Systems 12*. MIT Press, 2000.

[8] L. Kocsis and C. Szepesvari. Bandit based Monte-Carlo planning. In *15th European Conference on Machine Learning*, pages 282–293, 2006.

[9] H. Kurniawati, D. Hsu, and W. Lee. SARSOP: Efficient point-based POMDP planning by approximating optimally reachable belief spaces. In *Robotics: Science and Systems*, 2008.

[10] R. Lorentz. Amazons discover Monte-Carlo. In *Computers and Games*, pages 13–24, 2008.

[11] S. Ong, S. Png, D. Hsu, and W. Lee. POMDPs for robotic tasks with mixed observability. In *Robotics: Science and Systems*, 2009.

[12] J. Pineau, G. Gordon, and S. Thrun. Anytime point-based approximations for large POMDPs. *Journal of Artificial Intelligence Research*, 27:335–380, 2006.

[13] S. Ross, J. Pineau, S. Paquet, and B. Chaib-draa. Online planning algorithms for pomdps. *Journal of Artificial Intelligence Research*, 32:663–704, 2008.

[14] T. Smith and R. Simmons. Heuristic search value iteration for pomdps. In *20th conference on Uncertainty in Artificial Intelligence*, 2004.

[15] G. Tesauro and G. Galperin. Online policy improvement using Monte-Carlo search. In *Advances in Neural Information Processing 9*, pages 1068–1074, 1996.

